# Hierarchical Distributed Representations for Statistical Language Modeling

**John Blitzer, Kilian Q. Weinberger, Lawrence K. Saul, and Fernando C. N. Pereira**
Department of Computer and Information Science, University of Pennsylvania
Levine Hall, 3330 Walnut Street, Philadelphia, PA 19104
`{blitzer,kilianw,lsaul,pereira}@cis.upenn.edu`

## Abstract

Statistical language models estimate the probability of a word occurring in a given context. The most common language models rely on a discrete enumeration of predictive contexts (e.g., $n$-grams) and consequently fail to capture and exploit statistical regularities across these contexts. In this paper, we show how to learn hierarchical, distributed representations of word contexts that maximize the predictive value of a statistical language model. The representations are initialized by unsupervised algorithms for linear and nonlinear dimensionality reduction [14], then fed as input into a hierarchical mixture of experts, where each expert is a multinomial distribution over predicted words [12]. While the distributed representations in our model are inspired by the neural probabilistic language model of Bengio *et al.* [2, 3], our particular architecture enables us to work with significantly larger vocabularies and training corpora. For example, on a large-scale bigram modeling task involving a sixty thousand word vocabulary and a training corpus of three million sentences, we demonstrate consistent improvement over class-based bigram models [10, 13]. We also discuss extensions of our approach to longer multiword contexts.

## 1 Introduction

Statistical language models are essential components of natural language systems for human-computer interaction. They play a central role in automatic speech recognition [11], machine translation [5], statistical parsing [8], and information retrieval [15]. These models estimate the probability that a word will occur in a given context, where in general a context specifies a relationship to one or more words that have already been observed. The simplest, most studied case is that of $n$-gram language modeling, where each word is predicted from the preceding $n-1$ words. The main problem in building these models is that the vast majority of word combinations occur very infrequently, making it difficult to estimate accurate probabilities of words in most contexts.

Researchers in statistical language modeling have developed a variety of smoothing techniques to alleviate this problem of data sparseness. Most smoothing methods are based on simple back-off formulas or interpolation schemes that discount the probability of observed events and assign the "leftover" probability mass to events unseen in training [7]. Unfortunately, these methods do not typically represent or take advantage of statistical regularities

among contexts. One expects the probabilities of rare or unseen events in one context to be related to their probabilities in statistically similar contexts. Thus, it should be possible to estimate more accurate probabilities by exploiting these regularities.

Several approaches have been suggested for sharing statistical information across contexts. The aggregate Markov model (AMM) of Saul and Pereira [13] (also discussed by Hofmann and Puzicha [10] as a special case of the aspect model) factors the conditional probability table of a word given its context by a latent variable representing context "classes". However, this latent variable approach is difficult to generalize to multiword contexts, as the size of the conditional probability table for class given context grows exponentially with the context length.

The neural probabilistic language model (NPLM) of Bengio *et al.* [2, 3] achieved significant improvements over state-of-the-art smoothed $n$-gram models [6]. The NPLM encodes contexts as low-dimensional continuous vectors. These are fed to a multilayer neural network that outputs a probability distribution over words. The low-dimensional vectors and the parameters of the network are trained simultaneously to minimize the perplexity of the language model. This model has no difficulty encoding multiword contexts, but its training and application are very costly because of the need to compute a separate normalization for the conditional probabilities associated to each context.

In this paper, we introduce and evaluate a statistical language model that combines the advantages of the AMM and NPLM. Like the NPLM, it can be used for multiword contexts, and like the AMM it avoids per-context normalization. In our model, contexts are represented as low-dimensional real vectors initialized by unsupervised algorithms for dimensionality reduction [14]. The probabilities of words given contexts are represented by a hierarchical mixture of experts (HME) [12], where each expert is a multinomial distribution over predicted words. This tree-structured mixture model allows a rich dependency on context without expensive per-context normalization. Proper initialization of the distributed representations is crucial; in particular, we find that initializations from the results of linear and nonlinear dimensionality reduction algorithms lead to better models (with significantly lower test perplexities) than random initialization.

In practice our model is several orders of magnitude faster to train and apply than the NPLM, enabling us to work with larger vocabularies and training corpora. We present results on a large-scale bigram modeling task, showing that our model also leads to significant improvements over comparable AMMs.

## 2 Distributed representations of words

Natural language has complex, multidimensional semantics. As a trivial example, consider the following four sentences:

| The vase broke. | The vase contains water. |
| --- | --- |
| The window broke. | The window contains water. |

The bottom right sentence is syntactically valid but semantically meaningless. As shown by the table, a two-bit distributed representation of the words "vase" and "window" suffices to express that a vase is both a container and breakable, while a window is breakable but cannot be a container. More generally, we expect low dimensional *continuous* representations of words to be even more effective at capturing semantic regularities.

Distributed representations of words can be derived in several ways. In a given corpus of text, for example, consider the matrix of bigram counts whose element $C_{ij}$ records the number of times that word $w_j$ follows word $w_i$. Further, let $p_{ij} = C_{ij} / \sum_k C_{ik}$ denote the conditional frequencies derived from these counts, and let $\vec{p_i}$ denote the $V$-dimensional

frequency vector with elements $p_{ij}$, where $V$ is the vocabulary size. Note that the vectors $\vec{p}_i$ themselves provide a distributed representation of the words $w_i$ in the corpus. For large vocabularies and training corpora, however, this is an extremely unwieldy representation, tantamount to storing the full matrix of bigram counts. Thus, it is natural to seek a lower dimensional representation that captures the same information. To this end, we need to map each vector $\vec{p}_i$ to some $d$-dimensional vector $\vec{x}_i$, with $d \ll V$. We consider two methods in dimensionality reduction for this problem. The results from these methods are then used to initialize the HME architecture in the next section.

## 2.1  Linear dimensionality reduction

The simplest form of dimensionality reduction is principal component analysis (PCA). PCA computes a linear projection of the frequency vectors $\vec{p}_i$ into the low dimensional subspace that maximizes their variance. The variance-maximizing subspace of dimensionality $d$ is spanned by the top $d$ eigenvectors of the frequency vector covariance matrix. The eigenvalues of the covariance matrix measure the variance captured by each axis of the subspace. The effect of PCA can also be understood as a translation and rotation of the frequency vectors $\vec{p}_i$, followed by a truncation that preserves only their first $d$ elements.

## 2.2  Nonlinear dimensionality reduction

Intuitively, we would like to map the vectors $\vec{p}_i$ into a low dimensional space where semantically similar words remain close together and semantically dissimilar words are far apart. Can we find a nonlinear mapping that does this better than PCA? Weinberger *et al.* recently proposed a new solution to this problem based on semidefinite programming [14].

Let $\vec{x}_i$ denote the image of $\vec{p}_i$ under this mapping. The mapping is discovered by first learning the $V \times V$ matrix of Euclidean squared distances [1] given by $D_{ij} = |\vec{x}_i - \vec{x}_j|^2$. This is done by balancing two competing goals: (i) to co-locate semantically similar words, and (ii) to separate semantically dissimilar words. The first goal is achieved by fixing the distances between words with similar frequency vectors to their original values. In particular, if $\vec{p_j}$ and $\vec{p_k}$ lie within some small neighborhood of each other, then the corresponding element $D_{jk}$ in the distance matrix is fixed to the value $|\vec{p}_j - \vec{p}_k|^2$. The second goal is achieved by maximizing the sum of pairwise squared distances $\Sigma_{ij} D_{ij}$. Thus, we push the words in the vocabulary as far apart as possible subject to the constraint that the distances between semantically similar words do not change.

The only freedom in this optimization is the criterion for judging that two words are semantically similar. In practice, we adopt a simple criterion such as $k$-nearest neighbors in the space of frequency vectors $\vec{p}_i$ and choose $k$ as small as possible so that the resulting neighborhood graph is connected [14].

The optimization is performed over the space of Euclidean squared distance matrices [1]. Necessary and sufficient conditions for the matrix $D$ to be interpretable as a Euclidean squared distance matrix are that $D$ is symmetric and that the Gram matrix[1] derived from $G = -\frac{1}{2} H D H^T$ is semipositive definite, where $H = I - \frac{1}{V} \mathbf{1}\mathbf{1}^{\mathbf{T}}$. The optimization can thus be formulated as the semidefinite programming problem:

---

**Maximize $\Sigma_{ij} D_{ij}$ subject to: (i) $D^T = D$, (ii) $-\frac{1}{2} H D H \succeq 0$, and (iii) $D_{ij} = |\vec{p}_i - \vec{p}_j|^2$ for all neighboring vectors $\vec{p}_i$ and $\vec{p}_j$.**

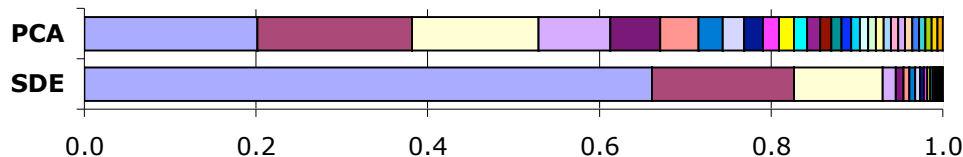

Figure 1: Eigenvalues from principal component analysis (PCA) and semidefinite embedding (SDE), applied to bigram distributions of the 2000 most frequently occuring words in the corpus. The eigenvalues, shown normalized by their sum, measure the relative variance captured by individual dimensions.

The optimization is convex, and its global maximum can be computed in polynomial time [4]. The optimization here differs slightly from the one used by Weinberger *et al.* [14] in that here we only preserve local distances, as opposed to local distances and angles.

After computing the matrix $D_{ij}$ by semidefinite programming, a low dimensional embedding $\vec{x}_i$ is obtained by metric multidimensional scaling [1, 9, 14]. The top eigenvalues of the Gram matrix measure the variance captured by the leading dimensions of this embedding. Thus, one can compare the eigenvalue spectra from this method and PCA to ascertain if the variance of the nonlinear embedding is concentrated in fewer dimensions. We refer to this method of nonlinear dimensionality reduction as semidefinite embedding (SDE). Fig. 1 compares the eigenvalue spectra of PCA and SDE applied to the 2000 most frequent words[2] in the corpus described in section 4. The figure shows that the nonlinear embedding by SDE concentrates its variance in many fewer dimensions than the linear embedding by PCA. Indeed, Fig. 2 shows that even the first two dimensions of the nonlinear embedding preserve the neighboring relationships of many words that are semantically similar. By contrast, the analogous plot generated by PCA (not shown) reveals no such structure.

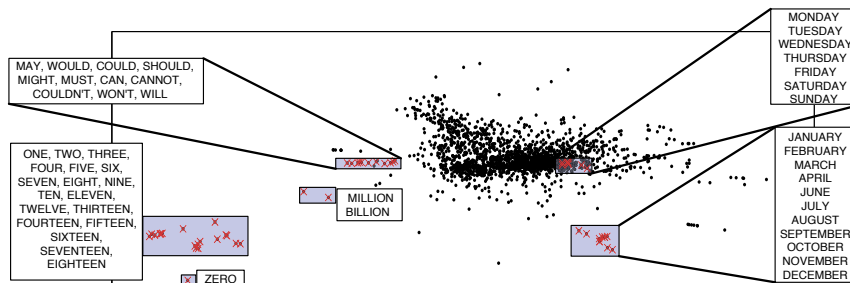

Figure 2: Projection of the normalized bigram counts of the 2000 most frequent words onto the first two dimensions of the nonlinear embedding obtained by semidefinite programming. Note that semantically meaningful neighborhoods are preserved, despite the massive dimensionality reduction from $V = 60000$ to $d = 2$.

## 3  Hierarchical mixture of experts

The model we use to compute the probability that word $w'$ follows word $w$ is known as a hierarchical mixture of experts (HME) [12]. HMEs are fully probabilistic models, making them ideally suited to the task of statistical language modeling. Furthermore, like multi-layer neural networks they can parameterize complex, nonlinear functions of their input.

Figure 3 depicts a simple, two-layer HME. HMEs are tree-structured mixture models in which the mixture components are "experts" that lie at the leaves of the tree. The interior nodes of the tree perform binary logistic regressions on the input vector to the HME, and the mixing weight for a leaf is computed by multiplying the probabilities of each branch (left or right) along the path to that leaf. In our model, the input vector $\vec{x}$ is a function of the context word $w$, and the expert at each leaf specifies a multinomial distribution over the predicted word $w'$. Letting $\pi$ denote a path through the tree from root to leaf, the HME computes the probability of a word $w'$ conditioned on a context word $w$ as

$$\Pr(w'|w) = \sum_{\pi} \Pr(\pi|\vec{x}(w)) \cdot \Pr(w'|\pi). \tag{1}$$

We can compute the maximum likelihood parameters for the HME using an Expectation-Maximization (EM) algorithm [12]. The E-step involves computing the posterior probability over paths $\Pr(\pi|w, w')$ for each observed bigram in the training corpus. This can be done by a recursive pass through the tree. In the M-step, we must maximize the EM auxiliary function with respect to the parameters of the logistic regressions and multinomial leaves as well as the input vectors $\vec{x}(w)$. The logistic regressions in the tree decouple and can be optimized separately by Newton's method, while the multinomial leaves have a simple closed-form update. Though the input vectors are shared across all logistic regressions in the tree, we can compute their gradients and hessians in one recursive pass and update them by Newton's method as well.

The EM algorithm for HMEs converges to a local maximum of the log-likelihood, or equivalently, a local minimum of the training perplexity

$$\mathcal{P}_{\text{train}} = \left\{ \prod_{ij} \Pr(w_j|w_i)^{C_{ij}} \right\}^{-\frac{1}{C}}, \tag{2}$$

where $C = \sum_{ij} C_{ij}$ is the total number of observed bigrams in the training corpus. The algorithm is sensitive to the choice of initialization; in particular, as we show in the next

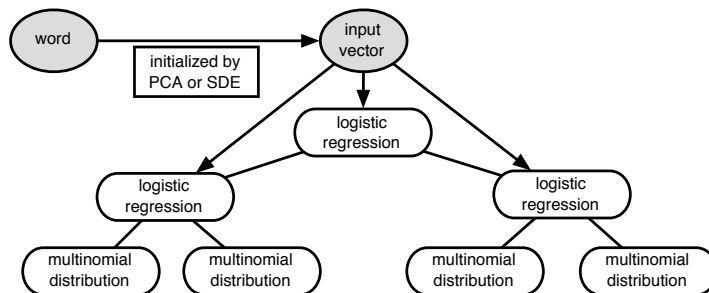

Figure 3: Two-layer HME for bigram modeling. Words are mapped to input vectors; probabilities of next words are computed by summing over paths through the tree. The mapping from words to input vectors is initialized by dimensionality reduction of bigram counts.

| $\mathcal{P}_{\text{test}}$ | $d$ | | | |
|---|---|---|---|---|
| **init** | 4 | 8 | 12 | 16 |
| random | 468 | 407 | 378 | 373 |
| PCA | 406 | 364 | 362 | 351 |
| SDE | 385 | 361 | 360 | 355 |

Table 1: Test perplexities of HMEs with different input dimensionalities and initializations.

| $\mathcal{P}_{\text{test}}$ | $d$ | | | |
|---|---|---|---|---|
| $m$ | 4 | 8 | 12 | 16 |
| 8 | 435 | 429 | 426 | 428 |
| 16 | 385 | 361 | 360 | 355 |
| 32 | 350 | 328 | 320 | 317 |
| 64 | 336 | 308 | 298 | 294 |

Table 2: Test perplexities of HMEs with different input dimensionalities and numbers of leaves.

section, initialization of the input vectors by PCA or SDE leads to significantly better models than random initialization. We initialized the logistic regressions in the HME to split the input vectors recursively along their dimensions of greatest variance. The multinomial distributions at leaf nodes were initialized by uniform distributions.

For an HME with $m$ multinomial leaves and $d$-dimensional input vectors, the number of parameters scales as $O(Vd + Vm + dm)$. The resulting model can be therefore be much more compact than a full bigram model over $V$ words.

# 4 Results

We evaluated our models on the ARPA North American Business News (NAB) corpus. Our training set contained 78 million words from a 60,000 word vocabulary. In the interest of speed, we truncated the lowest-count bigrams from our training set. This left us with a training set consisting of 1.7 million unique bigrams. The test set, untruncated, had 13 million words resulting in 2.1 million unique bigrams.

## 4.1 Empirical evaluation

Table 1 reports the test perplexities of several HMEs whose input vectors were initialized in different ways. The number of mixture components (i.e., leaves of the HME) was fixed at $m = 16$. In all cases, the inputs initialized by PCA and SDE significantly outperformed random initialization. PCA and SDE initialization performed equally well for all but the lowest-dimensional inputs. Here SDE outperformed PCA, most likely because the first few eigenvectors of SDE capture more variance in the bigram counts than those of PCA (see Figure 1).

Table 2 reports the test perplexities of several HMEs initialized by SDE, but with varying input dimensionality ($d$) and numbers of leaves ($m$). Perplexity decreases with increasing tree depth and input dimensionality, but increasing the dimensionality beyond $d = 8$ does not appear to give much gain.

## 4.2 Comparison to a class-based bigram model

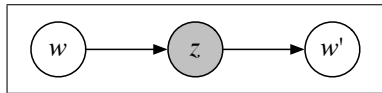

Figure 4: Belief network for AMM.

We obtained baseline results from an AMM [13] trained on the same corpus. The model (Figure 4) has the form

$$\Pr(w'|w) = \sum_z \Pr(z|w) \cdot \Pr(w'|z). \quad (3)$$

The number of estimated parameters in AMMs scales as $2 \cdot |Z| \cdot V$, where $|Z|$ is the size of the latent variable (i.e., number of classes) and $V$ is the number of words in the vocabulary.

| parameters (*1000) | $\mathcal{P}_{\text{test}}(\text{AMM})$ | $\mathcal{P}_{\text{test}}(\text{HME})$ | improvement |
|---|---|---|---|
| 960 | 456 | 429 | 6% |
| 1440 | 414 | 361 | 13% |
| 2400 | 353 | 328 | 7% |
| 4320 | 310 | 308 | 1% |

Table 3: Test perplexities of HMEs and AMMs with roughly equal parameter counts.

Table 3 compares the test perplexities of several HMEs and AMMs with similar numbers of parameters. All these HMEs had $d = 8$ inputs initialized by SDE. In all cases, the HMEs match or outperform the AMMs. The performance is nearly equal for the larger models, which may be explained by the fact that most of the parameters of the larger HMEs come from the multinomial leaves, not from the distributed inputs.

### 4.3 Comparison to NPLM

The most successful large-scale application of distributed representations to language modeling is the NPLM of Bengio *et al.* [2, 3], which in part inspired our work. We now compare the main aspects of the two models.

| $\tau$ | | $d$ | | |
|---|---|---|---|---|
| $m$ | 4 | 8 | 12 | 16 |
| 8 | 1 | 1 | 1 | 1 |
| 16 | 2 | 2 | 2 | 2 |
| 32 | 4 | 4 | 4 | 4 |
| 64 | 9 | 10 | 10 | 10 |

Table 4: Training times $\tau$ in hours for HMEs with $m$ leaves.

The NPLM uses softmax to compute the probability of a word $w'$ given its context, thus requiring a separate normalization for each context. Estimating the parameters of this softmax requires $O(V)$ computation per observed context and accounts for almost all of the computational resources required by the model. Because of this, the NPLM vocabulary size was restricted to 18000 words, and even then it required more than 3 weeks using 40 CPUs to finish 5 epochs of training [2].

By contrast, our HMEs require $O(md)$ computation per observed bigram. As Table 4 shows, actual training times are rather insensitive to input dimensionality. This allowed us to use a $3.5\times$ larger vocabulary and a larger training corpus than were used for the NPLM, and still complete training our largest models in a matter of hours. Note that the numbers in Table 4 do not include the time to compute the initial distributed representations by PCA (30 minutes) or SDE (3 days), but these computations do not need to be repeated for each trained model.

The second difference between our model and the NPLM is the choice of initialization. Bengio *et al.* [3] report negligible improvement from initializing the NPLM input vectors by singular value decomposition. By contrast, we found that initialization by PCA or SDE was essential for optimal performance of our models (Table 1).

Finally, the NPLM was applied to multiword contexts. We have not done these experiments yet, but our model extends naturally to multiword contexts, as we explain in the next section.

## 5   Discussion

In this paper, we have presented a statistical language model that exploits hierarchical distributed representations of word contexts. The model shares the advantages of the NPLM [2], but differs in its use of dimensionality reduction for effective parameter ini-

tialization and in the significant speedup provided by the HME architecture. We can consequently scale our models to larger training corpora and vocabularies. We have also demonstrated that our models consistently match or outperform a baseline class-based bigram model.

The class-based bigram model is nearly as effective as the HME, but it has the major drawback that there is no straightforward way to extend it to multiword contexts without exploding its parameter count. Like the NPLM, however, the HME can be easily extended. We can form an input vector for a multiword history $(w_1, w_2)$ simply by concatenating the vectors $\vec{x}(w_1)$ and $\vec{x}(w_2)$. The parameters of the corresponding HME can be learned by an EM algorithm similar to the one in this paper. Initialization from dimensionality reduction is also straightforward: we can compute the low dimensional representation for each word separately. We are actively pursuing these ideas to train models with hierarchical distributed representations of multiword contexts.

## Footnotes

[1]Assuming without loss of generality that the vectors $\vec{x}_i$ are centered on the origin, the dot products $G_{ij} = \vec{x}_i \cdot \vec{x}_j$ are related to the pairwise squared distances $D_{ij} = |\vec{x}_i - \vec{x}_j|^2$ as stated above.

[2]Though convex, the optimization over distance matrices for SDE is prohibitively expensive for large matrices. For the results in this paper—on the corpus described in section 4—we solved the semidefinite program in this section to embed the 2000 most frequent words in the corpus, then used a greedy incremental solver to embed the remaining 58000 words in the vocabulary. Details of this incremental solver will be given elsewhere. Though not the main point of this paper, the nonlinear embedding of $V = 60000$ words is to our knowledge one of the largest applications of recently developed spectral methods for nonlinear dimensionality reduction [9, 14].

## References

[1] A. Y. Alfakih, A. Khandani, and H. Wolkowicz. Solving Euclidean distance matrix completion problems via semidefinite programming. *Computational Optimization Applications*, 12(1-3):13–30, 1999.

[2] Y. Bengio, R. Ducharme, P. Vincent, and C. Janvin. A neural probabilistic language model. *Journal of Machine Learning Research*, 3:1137–1155, 2003.

[3] Y. Bengio, R. Ducharme, P. Vincent, and C. Jauvin. A neural probabilistic language model. In T. K. Leen, T. G. Dietterich, and V. Tresp, editors, *Advances in Neural Information Processing Systems*, volume 13, Cambridge, MA, 2001. MIT Press.

[4] D. B. Borchers. CSDP, a C library for semidefinite programming. *Optimization Methods and Software*, 11(1):613–623, 1999.

[5] P. Brown, S. D. Pietra, V. D. Pietra, and R. Mercer. The mathematics of statistical machine translation: parameter estimation. *Computational Linguistics*, 19(2):263–311, 1991.

[6] P. F. Brown, V. J. D. Pietra, P. V. deSouza, J. C. Lai, and R. L. Mercer. Class-based n-gram models of natural language. *Computational Linguistics*, 18(4):467–479, 1992.

[7] S. Chen and J. Goodman. An empirical study of smoothing techniques for language modeling. In *Proceedings of the 34th Annual Meeting of the ACL*, pages 310–318, 1996.

[8] M. Collins. Three generative, lexicalised models for statistical parsing. In *Proceedings of the 35th Annual Meeting of the Association for Computational Linguistics*, 1997.

[9] J. Ham, D. D. Lee, S. Mika, and B. Schölkopf. A kernel view of the dimensionality reduction of manifolds. In *Proceedings of the Twenty First International Conference on Machine Learning (ICML-04)*, Banff, Canada, 2004.

[10] T. Hofmann and J. Puzicha. Statistical models for co-occurrence and histogram data. In *Proceedings of the International Conference Pattern Recognition*, pages 192–194, 1998.

[11] F. Jelinek. *Statistical Methods for Speech Recognition*. MIT Press, 1997.

[12] M. I. Jordan and R. A. Jacobs. Hierarchical mixtures of experts and the EM algorithm. *Neural Computation*, 6:181–214, 1994.

[13] L. K. Saul and F. C. N. Pereira. Aggregate and mixed-order Markov models for statistical language processing. In C. Cardie and R. Weischedel, editors, *Proceedings of the Second Conference on Empirical Methods in Natural Language Processing (EMNLP-97)*, pages 81–89, New Providence, RI, 1997.

[14] K. Q. Weinberger, F. Sha, and L. K. Saul. Learning a kernel matrix for nonlinear dimensionality reduction. In *Proceedings of the Twenty First International Confernence on Machine Learning (ICML-04)*, Banff, Canada, 2004.

[15] C. Zhai and J. Lafferty. A study of smoothing methods for language models applied to information retrieval. *ACM Transactions on Information Systems*, 22(2):179–214, 2004.
